# Fast Kernels for String and Tree Matching

**S. V. N. Vishwanathan**
Dept. of Comp. Sci. & Automation
Indian Institute of Science
Bangalore, 560012, India
vishy@csa.iisc.ernet.in

**Alexander J. Smola**
Machine Learning Group, RSISE
Australian National University
Canberra, ACT 0200, Australia
Alex.Smola@anu.edu.au

## Abstract

In this paper we present a new algorithm suitable for matching discrete objects such as strings and trees in linear time, thus obviating dynamic programming with quadratic time complexity. Furthermore, prediction cost in many cases can be reduced to linear cost in the length of the sequence to be classified, regardless of the number of support vectors. This improvement on the currently available algorithms makes string kernels a viable alternative for the practitioner.

## 1 Introduction

Many problems in machine learning require the classifier to work with a set of discrete examples. Common examples include biological sequence analysis where data is represented as strings [4] and Natural Language Processing (NLP) where the data is in the form a parse tree [3]. In order to apply kernel methods one defines a measure of similarity between discrete structures via a feature map $\phi : \mathcal{X} \to \mathcal{H}_k$.

Here $\mathcal{X}$ is the set of discrete structures (eg. the set of all parse trees of a language) and $\mathcal{H}_K$ is a Hilbert space. Furthermore, dot products then lead to kernels

$$k(x, x') = \langle \phi(x), \phi(x') \rangle \tag{1}$$

where $x, x' \in \mathcal{X}$. The success of a kernel method employing $k$ depends both on the *faithful representation* of discrete data and an *efficient means of computing $k$*.

This paper presents a means of computing kernels on strings [15, 7, 12] and trees [3] in *linear time* in the size of the arguments, regardless of the weighting that is associated with any of the terms, plus linear time complexity for prediction, regardless of the number of support vectors. This is a significant improvement, since the so-far fastest methods [8, 3] rely on dynamic programming which incurs a quadratic cost in the length of the argument. Note that the method we present here is far more general than strings and trees, and it can be applied to finite state machines, formal languages, automata, etc. to define new kernels [14]. However for the scope of the current paper we limit ourselves to a fast means of computing extensions of the kernels of [15, 3, 12].

In a nutshell our idea works as follows: assume we have a kernel $k(x, x') = \sum_{i \in I} \phi_i(x)\phi_i(x')$, where the index set $I$ may be large, yet the number of nonzero entries is small in comparison to $|I|$. Then an efficient way of computing $k$ is to sort the set of nonzero entries $\phi(x)$ and $\phi(x')$ beforehand and count only matching non-zeros. This is similar to the dot-product of sparse vectors in numerical mathematics. As long as the sorting is done in an intelligent manner, the cost of computing $k$ is linear in the sum of non-zeros entries combined. In order to use this idea for matching strings (which have a

quadratically increasing number of substrings) and trees (which can be transformed into strings) efficient sorting is realized by the compression of the set of all substrings into a suffix tree. Moreover, dictionary keeping allows us to use arbitrary weightings for each of the substrings and still compute the kernels in linear time.

## 2    String Kernels

We begin by introducing some notation. Let $\mathcal{A}$ be a finite set which we call the *alphabet*. The elements of $\mathcal{A}$ are *characters*. Let $ be a sentinel character such that $ \notin \mathcal{A}$. Any $x \in \mathcal{A}^k$ for $k = 0, 1, 2 \ldots$ is called a *string*. The empty string is denoted by $\epsilon$ and $\mathcal{A}^*$ represents the set of all non empty strings defined over the alphabet $\mathcal{A}$.

In the following we will use $s, t, u, v, w, x, y, z \in \mathcal{A}^*$ to denote strings and $a, b, c \in \mathcal{A}$ to denote characters. $|x|$ denotes the length of $x$, $uv \in \mathcal{A}^*$ the concatenation of two strings $u, v$ and $au$ the concatenation of a character and a string. We use $x[i : j]$ with $1 \leq i \leq j \leq |x|$ to denote the substring of $x$ between locations $i$ and $j$ (both inclusive). If $x = uvw$ for some (possibly empty) $u, v, w$, then $u$ is called a *prefix* of $x$ while $v$ is called a substring (also denoted by $v \sqsubseteq x$) and $w$ is called a *suffix* of $x$. Finally, $\text{num}_y(x)$ denotes the number of occurrences of $y$ in $x$. The type of kernels we will be studying are defined by

$$k(x, x') := \sum_{s \sqsubseteq x, s' \sqsubseteq x'} w_s \delta_{s, s'} = \sum_{s \in \mathcal{A}^*} \text{num}_s(x) \, \text{num}_s(x') w_s. \qquad (2)$$

That is, we count the number of occurrences of every string $s$ in both $x$ and $x'$ and weight it by $w_s$, where the latter may be a weight chosen *a priori* or after seeing data, e.g., for inverse document frequency counting [11]. This includes a large number of special cases:

- Setting $w_s = 0$ for all $|s| > 1$ yields the bag-of-characters kernel, counting simply single characters.
- The bag-of-words kernel is generated by requiring $s$ to be bounded by whitespace.
- Setting $w_s = 0$ for all $|s| > n$ yields limited range correlations of length $n$.
- The *k-spectrum* kernel takes into account substrings of length $k$ [12]. It is achieved by setting $w_s = 0$ for all $|s| \neq k$.
- TFIDF weights are achieved by first creating a (compressed) list of all $s$ including frequencies of occurrence, and subsequently rescaling $w_s$ accordingly.

All these kernels can be computed efficiently via the construction of suffix-trees, as we will see in the following sections. However, before we do so, let us turn to trees. The latter are important for two reasons: first since the *suffix tree* representation of a string will be used to compute kernels efficiently, and secondly, since we may wish to compute kernels on trees, which will be carried out by reducing trees to strings and then applying a string-kernel.

## 3    Tree Kernels

A tree is defined as a connected directed graph with no cycles. A node with no children is referred to as a *leaf*. A subtree rooted at node $n$ is denoted as $T_n$ and $t \models T$ is used to indicate that $t$ is a subtree of $T$. If a set of nodes in the tree along with the corresponding edges forms a tree then we define it to be a *subset tree*. If every node $n$ of the tree contains a label, denoted by $\text{label}(n)$, then the tree is called an *labeled* tree. If only the leaf nodes contain labels then the tree is called an *leaf-labeled* tree. Kernels on trees can be defined by defining kernels on matching subset trees as proposed by [3] or (more restrictively) by defining kernels on matching subtrees. In the latter case we have

$$k(T, T') = \sum_{t \models T, t' \models T'} w_t \delta_{t, t'}. \qquad (3)$$

**Ordering Trees**    An *ordered* tree is one in which the child nodes of every node are ordered as per the ordering defined on the node labels. Unless there is a specific inherent order on the trees we are given (which is, e.g., the case for parse-trees), the representation of trees is

not unique. For instance, the following two unlabeled trees are equivalent and can obtained from each other by reordering the nodes.

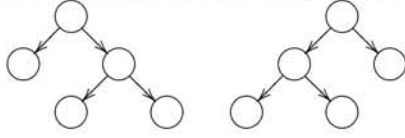

**Figure 1:** Two equivalent trees

To order trees we assume that a lexicographic order is associated with the labels if they exist. Furthermore, we assume that the additional symbols '$[$', '$]$' satisfy '$[$'$<$'$]$', and that '$]$', '$[$'$<$ label$(n)$ for all labels. We will use these symbols to define tags for each node as follows:

- For an unlabeled leaf $n$ define tag$(n) :=$ [ ].
- For a labeled leaf $n$ define tag$(n) :=$ [ label$(n)$].
- For an unlabeled node $n$ with children $n_1, \ldots, n_c$ sort the tags of the children in lexicographical order such that tag$(n_i) \le$ tag$(n_j)$ if $i < j$ and define

$$\text{tag}(n) = [\ \text{tag}(n_1)\,\text{tag}(n_2)\ldots\text{tag}(n_c)\,].$$

- For a labeled node perform the same operations as above and set

$$\text{tag}(n) = [\ \text{label}(n)\,\text{tag}(n_1)\,\text{tag}(n_2)\ldots\text{tag}(n_c)\,].$$

For instance, the root nodes of both trees depicted above would be encoded as [[][[][]]]. We now prove that the tag of the root node, indeed, is a unique identifier and that it can be constructed in log linear time.

**Theorem 1** *Denote by $T$ a binary tree with $l$ nodes and let $\lambda$ be the maximum length of a label. Then the following properties hold for the tag of the root node:*

1. *tag(root) can be computed in $(\lambda + 2)(l \log_2 l)$ time and linear storage in $l$.*
2. *Substrings $s$ of tag(root) starting with '$[$' and ending with a balanced '$]$' correspond to subtrees $T'$ of $T$ where $s$ is the tag on $T'$.*
3. *Arbitrary substrings $s$ of tag(root) correspond to subset trees $T'$ of $T$.*
4. *tag(root) is invariant under permutations of the leaves and allows the reconstruction of an unique element of the equivalence class (under permutation).*

**Proof** We prove claim 1 by induction. The tag of a leaf can be constructed in constant time by storing [, ], and a pointer to the label of the leaf (if it exists), that is in 3 operations. Next assume that we are at node $n$, with children $n_1, n_2$. Let $T_n$ contain $l_n$ nodes and $T_{n_1}$ and $T_{n_2}$ contain $l_1, l_2$ nodes respectively. By our induction assumption we can construct the tag for $n_1$ and $n_2$ in $(\lambda + 2)(l_1 \log_2 l_1)$ and $(\lambda + 2)(l_2 \log_2 l_2)$ time respectively. Comparing the tags of $n_1$ and $n_2$ costs at most $(\lambda + 2)\min(l_1, l_2)$ operations and the tag itself can be constructed in constant time and linear space by manipulating pointers. Without loss of generality we assume that $l_1 \le l_2$. Thus, the time required to construct tag$(n)$ (normalized by $\lambda + 2$) is

$$l_1(\log_2 l_1 + 1) + l_2 \log_2(l_2) = l_1 \log_2(2l_1) + l_2 \log_2(l_2) \le l_n \log_2(l_n). \qquad (4)$$

One way of visualizing our ordering is by imagining that we perform a DFS (depth first search) on the tree $T$ and emit a $'[\,'$ followed by the label on the node, when we visit a node for the first time and a $']\,'$ when we leave a node for the last time. It is clear that a *balanced* substring $s$ of tag(root) is emitted only when the corresponding DFS on $T'$ is completed. This proves claim 2.

We can emit a substring of tag(root) only if we can perform a DFS on the corresponding set of nodes. This implies that these nodes constitute a tree and hence by definition are subset trees of $T$. This proves claim 3.

Since leaf nodes do not have children their tag is clearly invariant under permutation. For an internal node we perform lexicographic sorting on the tags of its children. This removes any dependence on permutations. This proves the invariance of tag(root) under permutations of the leaves. Concerning the reconstruction, we proceed as follows: each tag of a subtree starts with $'[\,'$ and ends in a balanced $']\,'$, hence we can strip the first [] pair from the tag,

take whatever is left outside brackets as the label of the root node, and repeat the procedure with the balanced [...] entries for the children of the root node. This will construct a tree with the same tag as $\mathrm{tag}(\mathrm{root})$, thus proving claim 4. ∎

An extension to trees with $d$ nodes is straightforward (the cost increases to $d \log_2 d$ of the original cost), yet the proof, in particular (4) becomes more technical without providing additional insight, hence we omit this generalization for brevity.

**Corollary 2** *Kernels on trees $T, T'$ can be computed via string kernels, if we use $\mathrm{tag}(T), \mathrm{tag}(T')$ as strings. If we require that only balanced [...] substrings have nonzero weight $w_s$ then we obtain the subtree matching kernel defined in (3).*

This reduces the problem of tree kernels to string kernels and all we need to show in the following is how the latter can be computed efficiently. For this purpose we need to introduce suffix trees.

## 4 Suffix Trees and Matching Statistics

**Definition** The suffix tree is a compacted trie that stores all suffixes of a given text string. We denote the suffix tree of the string $x$ by $S(x)$. Moreover, let $\mathrm{nodes}(S(x))$ be the set of all nodes of $S(x)$ and let $\mathrm{root}(S(x))$ be the root of $S(x)$. For a node $w$, $\mathrm{father}(w)$ denotes its parent, $T(w)$ denotes the subtree tree rooted at the node, $\mathrm{lvs}(w)$ denotes the number of leaves in the subtree and $\mathrm{path}(w) := w$ is the path from the root to the node. That is, we use the path $w$ from root to node as the label of the node $w$.

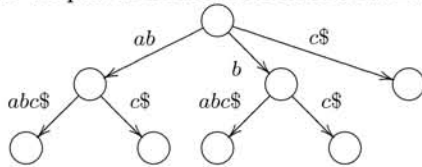

**Figure 2:** Suffix Tree of `ababc`

We denote by $\mathrm{words}(S(x))$ the set of all strings $w$ such that $wu \in \mathrm{nodes}(S(x))$ for some (possibly empty) string $u$, which means that $\mathrm{words}(S(x))$ is the set of all possible substrings of $x$. For every $t \in \mathrm{words}(S(x))$ we define $\mathrm{ceil}(t)$ as the node $w$ such that $w = tu$ and $u$ is the shortest (possibly empty) substring such that $w \in \mathrm{nodes}(S(x))$. Similarly, for every $t \in \mathrm{words}(S(x))$ we define $\mathrm{floor}(t)$ as the node $w$ such that $t = wu$ and $u$ is the shortest (possibly empty) substring such that $w \in \mathrm{nodes}(S(x))$. Given a string $t$ and a suffix tree $S(x)$, we can decide if $t \in \mathrm{words}(S(x))$ in $O(|t|)$ time by just walking down the corresponding edges of $S(x)$.

If the sentinel character $ is added to the string $x$ then it can be shown that for any $t \in \mathrm{words}(S(x))$, $\mathrm{lvs}(\mathrm{ceil}(t))$ gives us the number of occurrence of $t$ in $x$ [5]. The idea works as follows: all suffixes of $x$ starting with $t$ have to pass through $\mathrm{ceil}(t)$, hence we simply have to count the occurrences of the sentinel character, which can be found only in the leaves. Note that a simple depth first search (DFS) of $S(x)$ will enable us to calculate $\mathrm{lvs}(w)$ for each node in $S(x)$ in $O(|x|)$ time and space.

Let $aw$ be a node in $S(x)$, and $v$ be the longest suffix of $w$ such that $v \in \mathrm{nodes}(S(x))$. An unlabeled edge $aw \rightarrow v$ is called a suffix link in $S(x)$. A suffix link of the form $aw \rightarrow w$ is called *atomic*. It can be shown that all the suffix links in a suffix tree are atomic [5, Proposition 2.9]. We add suffix links to $S(x)$, to allow us to perform efficient string matching: suppose we found that $aw$ is a substring of $x$ by parsing the suffix tree $S(x)$. It is clear that $w$ is also a substring of $x$. If we want to locate the node corresponding to $w$, it would be wasteful to parse the tree again. Suffix links can help us locate this node in constant time. The suffix tree building algorithms make use of this property of suffix links to perform the construction in linear time. The suffix tree construction algorithm of [13] constructs the suffix tree and all such suffix links in *linear time*.

**Matching Statistics** Given strings $x, y$ with $|x| = n$ and $|y| = m$, the matching statistics of $x$ with respect to $y$ are defined by $v, c \in \mathbb{N}^n$, where $v_i$ is the length of the longest substring of $y$ matching a prefix of $x[i : n]$, $v_i := i + v_i - 1$, $c_i$ is a pointer to $\mathrm{ceil}(x[i : v_i])$ and $c_i$ is a pointer to $\mathrm{floor}(x[i : v_i])$ in $S(y)$. For an example see the table below.

| String | a | b | b | a |
|--------|-----|---|-------|----|
| $v_i$ | 2 | 1 | 2 | 1 |
| $c_i$ | ab | b | babc\$ | ab |

**Table 1:** Matching statistic of `abba` with respect to $S(\text{ababc})$.

For a given $y$ one can construct $v, c$ corresponding to $x$ in linear time. The key observation is that $v_{i+1} \geq v_i - 1$, since if $x[i : v_i]$ is a substring of $y$ then definitely $x[i + 1 : v_i]$ is also a substring of $y$. Besides this, the matching substring in $y$ that we find, *must* have $x[i+1 : v_i]$ as a prefix. The Matching Statistics algorithm [2] exploits this observation and uses it to cleverly walk down the suffix links of $S(y)$ in order to compute the matching statistics in $O(|x|)$ time.

More specifically, the algorithm works by maintaining a pointer $p_i = \text{floor}(x[i : v_i])$. It then finds $p_{i+1} = \text{floor}(x[i + 1 : v_i])$ by first walking down the suffix link of $p_i$ and then walking down the edges corresponding to the remaining portion of $x[i + 1 : v_i]$ until it reaches $\text{floor}(x[i + 1 : v_i])$. Now $v_{i+1}$ can be found easily by walking from $p_{i+1}$ along the edges of $S(y)$ that match the string $x[i + l : n]$, until we can go no further. The value of $v_1$ is found by simply walking down $S(y)$ to find the longest prefix of $x$ which matches a substring of $y$.

**Matching substrings**   Using $v$ and $c$ we can read off the number of matching substrings in $x$ and $y$. The useful observation here is that the only substrings which occur in both $x$ and $y$ are those which are prefixes of $x[i : v_i]$. The number of occurrences of a substring in $y$ can be found by $\text{lvs}(\text{ceil}(w))$ (see Section 4). The two lemmas below formalize this.

**Lemma 3** *$w$ is a substring of $x$ iff there is an $i$ such that $w$ is a prefix of $x[i : n]$. The number of occurrences of $w$ in $x$ can be calculated by finding all such $i$.*

**Lemma 4** *The set of matching substrings of $x$ and $y$ is the set of all prefixes of $x[i : v_i]$.*

**Proof**   Let $w$ be a substring of both $x$ and $y$. By above lemma there is an $i$ such that $w$ is a prefix of $x[i : n]$. Since $v_i$ is the length of the maximal prefix of $x[i : n]$ which is a substring in $y$, it follows that $v_i \geq |w|$. Hence $w$ must be a prefix of $x[i : v_i]$. ∎

## 5   Weights and Kernels

From the previous sections we know how to determine the set of all longest prefixes $x[i : v_i]$ of $x[i : n]$ in $y$ in linear time. The following theorem uses this information to compute kernels efficiently.

**Theorem 5** *Let $x$ and $y$ be strings and $c$ and $v$ be the matching statistics of $x$ with respect to $y$. Assume that*

$$W(y, t) = \sum_{s \in \text{prefix}(v)} w_{us} - w_u \text{ where } u = \text{floor}(t) \text{ and } t = uv. \tag{5}$$

*can be computed in constant time for any $t$. Then $k(x, y)$ can be computed in $O(|x| + |y|)$ time as*

$$k(x, y) = \sum_{i=1}^{|x|} \text{val}(x[i : v_i]) = \sum_{i=1}^{|x|} \text{val}(c_i) + \text{lvs}(\text{ceil}(x[i : v_i])) W(y, x[i : v_i]) \tag{6}$$

*where $\text{val}(t) := \text{lvs}(\text{ceil}(t)) \cdot W(y, t) + \text{val}(\text{floor}(t))$ and $\text{val}(\text{root}) := 0$.*

**Proof**   We first show that (6) can indeed be computed in linear time. We know that for $S(y)$ the number of leaves can be computed in linear time and likewise $c, v$. By assumption on $W(y, t)$ and by exploiting the recursive nature of $\text{val}(t)$ we can compute $W(y, \text{nodes}(i))$ for all the nodes of $S(y)$ by a simple top down procedure in $O(|y|)$ time.

Also, due to recursion, the second equality of (6) holds and we may compute each term in constant time by a simple lookup for $\text{val}(c_i)$ and computation of $W(y, x[i : v_i])$. Since we have $|x|$ terms, the whole procedure takes $O(|x|)$ time, which proves the $O(|x| + |y|)$ time complexity.

Now we prove that (6) really computes the kernel. We know from Lemma 4 that the sum in (2) can be decomposed into the sum over matches between $y$ and each of the prefixes

of $x[i : v_i]$ (this takes care of all the substrings in $x$ matching with $y$). This reduces the problem to showing that each term in the sum of (6) corresponds to the contribution of all prefixes of $x[i : v_i]$.

Assume we descend down the path $x[i : v_i]$ in $S(y)$ (e.g., for the string `bab` with respect to the tree of Figure 2 this would correspond to (root, b, bab)), then each of the prefixes $t$ along the path (e.g., (′ ′, b, ba, bab) for the example tree) occurs exactly as many times as lvs(ceil($t$)) does. In particular, prefixes ending on the same edge occur the same number of times. This allows us to bracket the sums efficiently, and $W(y, x)$ simply is the sum along an edge, starting from the ceiling of $x$ to $x$. Unwrapping val($x$) shows that this is simply the sum over the occurrences on the path of $x$, which proves our claim. ∎

So far, our claim hinges on the fact that $W(y, t)$ can be computed in constant time, which is far from obvious at first glance. We now show that this is a reasonable assumption in all practical cases.

**Length Dependent Weights**  If the weights $w_s$ depend only on $|s|$ we have $w_s = w_{|s|}$. Define $\omega_j := \sum_{i=1}^{j} w_j$ and compute its values beforehand up to $\omega_J$ where $J \geq |x|$ for all $x$. Then it follows that

$$W(y, t) = \sum_{j=|\text{ceil}(t)|}^{|t|} w_j - w_{|\text{floor}(t)|} = \omega_{|t|} - \omega_{|\text{floor}(t)|} \tag{7}$$

which can be computed in constant time. Examples of such weighting schemes are the kernels suggested by [15], where $w_i = \lambda^{-i}$, [7] where $w_i = 1$, and [10], where $w_i = \delta_{1i}$.

**Generic Weights**  In case of generic weights, we have several options: recall that one often will want to compute $m^2$ kernels $k(x, x')$, given $m$ strings $x \in X$. Hence we could build the suffix trees for $x_i$ beforehand and annotate each of the nodes and characters on the edges explicitly (at super-linear cost per string), which means that later, for the dot products, we will only need to perform table lookup of $W(x, x'(i : v_i))$.

However, there is an even more efficient mechanism, which can even deal with dynamic weights, depending on the relative frequency of occurrence of the substrings in all $x$. We can build a suffix tree $\Sigma$ of all strings in $X$. Again, this can be done in time linear in the total length of all the strings (simply consider the concatenation of all strings). It can be shown that for all $x$ and all $i$, $x[i : v_i]$ will be a node in this tree. Leaves-counting allows to compute these dynamic weights efficiently, since $\Sigma$ contains all the substrings.

For $W(x, x'(i : v_i))$ we make the simplifying assumption that $w_s = \phi(|s|) \cdot \phi(\text{freq}(s))$, that is, $w_s$ depends on length and frequency only. Now note that all the strings ending on the same edge in $\Sigma$ will have the same weights assigned to them. Hence, can rewrite (5) as

$$W(y, t) = \sum_{s \in \text{prefix}(t)} w_s - \sum_{s \in \text{prefix}(\text{floor}(t))} w_s = \phi(\text{freq}(t)) \sum_{i=|\text{floor}(t)|+1}^{|t|} \phi(i) \tag{8}$$

where $u = \text{floor}(t)$, $t = uv$ and $s \in \text{prefix}(v)$. By precomputing $\sum_i \phi(i)$ we can evaluate (8) in constant time.

The benefit of (8) is twofold: we can compute the weights of all the nodes of $\Sigma$ in time linear in the total length of strings in $X$. Secondly, for arbitrary $x$ we can compute $W(y, t)$ in constant time, thus allowing us to compute $k(x_i, x')$ in $O(|x_i| + |x'|)$ time.

**Linear Time Prediction**  Let $\mathcal{X}_s = \{x_1, x_2, \ldots, x_m\}$ be the set of support vectors. Recall that, for prediction in a Support Vector Machine we need to compute $f(x) = \sum_{i=1}^{m} \alpha_i k(x_i, x)$, which implies that we need to combine the contribution due to matching substrings from each one of the Support Vectors. We first construct $S(\mathcal{X}_s)$ in linear time by using the [1] algorithm. In $S(\mathcal{X}_s)$, we associate weight $\alpha_i$ with each leaf associated with the support vector $x_i$. For a node $v \in \text{nodes}(S(\mathcal{X}_s))$ we modify the definition of lvs($v$) as the sum of weights associated with the subtree rooted at node $v$. A straightforward application of the matching statistics algorithm of [2] shows that we can find the matching

statistics of $x$ with respect to all strings in $\mathfrak{X}_s$ in $O(|x|)$ time. Now Theorem 5, can be applied unchanged to compute $f(x)$. A detailed account and proof can be found in [14]. In summary, we can classify texts in linear time regardless of the size of the training set. This makes SVM for large-scale text categorization practically feasible. Similar modifications can also be applied for training SMO like algorithms on strings.

## 6 Experimental Results

For a proof of concept we tested our approach on a remote homology detection problem[1] [9] using Stafford Noble's SVM package[2] as the training algorithm. A length weighted kernel was used and we assigned weights $w_s = \lambda^{|s|}$ for all substring matches of length greater than 3 regardless of triplet boundaries. To evaluate performance we computed the $\mathrm{ROC}_{50}$ scores.[3]

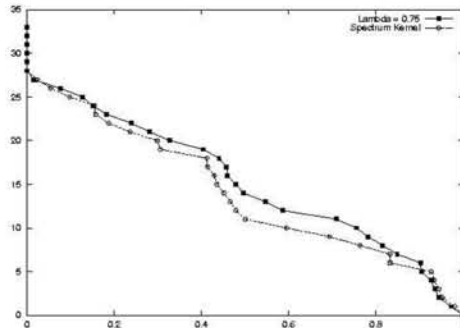

**Figure 3:** Total number of families for which an SVM classifier exceeds a $ROC_{50}$ score threshold.

Being a proof of concept, we did not try to tune the soft margin SVM parameters (the main point of the paper being the introduction of a novel means of evaluating string kernels efficiently rather than applications - a separate paper focusing on applications is in preparation). Table 3 contains the $\mathrm{ROC}_{50}$ scores for the spectrum kernel with $k = 3$ [12] and our string kernel with $\lambda = 0.75$. We tested with $\lambda \in \{0.25, 0.5, 0.75, 0.9\}$ and report the best results here. As can be seen our kernel outperforms the spectrum kernel on nearly every every family in the dataset.

It should be noted that this is the first method to allow users to specify weights rather arbitrarily for all possible lenghts of matching sequences and still be able to compute kernels at $O(|x| + |x'|)$ time, plus, to predict on new sequences at $O(|x|)$ time, once the set of support vectors is established.[4]

## 7 Conclusion

We have shown that string kernels need not come at a super-linear cost in SVMs and that prediction can be carried out at cost linear only in the length of the argument, thus providing optimal run-time behaviour. Furthermore the same algorithm can be applied to trees.

The methodology pointed out in our paper has several immediate extensions: for instance, we may consider coarsening levels for trees by removing some of the leaves. For not too-unbalanced trees (we assume that the tree shrinks at least by a constant factor at each coarsening) computation of the kernel over all coarsening levels can then be carried out at cost still linear in the overall size of the tree. The idea of coarsening can be extended to approximate string matching. If we remove characters, this amounts to the use of wildcards. Likewise, we can consider the strings generated by finite state machines and thereby compare the finite state machines themselves. This leads to kernels on automata and other dynamical systems. More details and extensions can be found in [14].

**Acknowledgments** We would like to thank Patrick Haffner, Daniela Pucci de Farias, and Bob Williamson for comments and suggestions. This research was supported by a grant of the Australian Research Council. SVNV thanks Trivium India Software and Netscaler Inc. for their support.

## Footnotes

[1]Details and data available at www.cse.ucsc.edu/research/compbio/discriminative.

[2]Available at www.cs.columbia.edu/compbio/svm.

[3]The $\mathrm{ROC}_{50}$ score [6, 12] is the area under the receiver operating characteristic curve (the plot of true positives as a function of false positives) up to the first 50 false positives. A score of 1 indicates perfect separation of positives from negatives, whereas a score of 0 indicates that none of the top 50 sequences selected by the algorithm were positives.

[4][12] obtain an $O(k|x|)$ algorithm in the (somewhat more restrictive) case of $w_s = \delta_k(|s|)$.

# References

[1] A. Amir, M. Farach, Z. Galil, R. Giancarlo, and K. Park. Dynamic dictionary matching. *Journal of Computer and System Science*, 49(2):208–222, October 1994.

[2] W. I. Chang and E. L. Lawler. Sublinear approximate sting matching and biological applications. *Algorithmica*, 12(4/5):327–344, 1994.

[3] M. Collins and N. Duffy. Convolution kernels for natural language. In T. G. Dietterich, S. Becker, and Z. Ghahramani, editors, *Advances in Neural Information Processing Systems 14*, Cambridge, MA, 2001. MIT Press.

[4] R. Durbin, S. Eddy, A. Krogh, and G. Mitchison. *Biological Sequence Analysis: Probabilistic models of proteins and nucleic acids*. Cambridge University Press, 1998.

[5] R. Giegerich and S. Kurtz. From Ukkonen to McCreight and Weiner: A unifying view of linear-time suffix tree construction. *Algorithmica*, 19(3):331–353, 1997.

[6] M. Gribskov and N. L. Robinson. Use of receiver operating characteristic (ROC) analysis to evaluate sequence matching. *Computers and Chemistry*, 20(1):25–33, 1996.

[7] D. Haussler. Convolutional kernels on discrete structures. Technical Report UCSC-CRL-99-10, Computer Science Department, UC Santa Cruz, 1999.

[8] R. Herbrich. *Learning Kernel Classifiers: Theory and Algorithms*. MIT Press, 2002.

[9] T. S. Jaakkola, M. Diekhans, and D. Haussler. A discriminative framework for detecting remote protein homologies. *Journal of Computational Biology*, 7:95–114, 2000.

[10] T. Joachims. Making large-scale SVM learning practical. In B. Schölkopf, C. J. C. Burges, and A. J. Smola, editors, *Advances in Kernel Methods—Support Vector Learning*, pages 169–184, Cambridge, MA, 1999. MIT Press.

[11] E. Leopold and J. Kindermann. Text categorization with support vector machines: How to represent text in input space? *Machine Learning*, 46(3):423–444, March 2002.

[12] C. Leslie, E. Eskin, and W. S. Noble. The spectrum kernel: A string kernel for SVM protein classification. In *Proceedings of the Pacific Symposium on Biocomputing*, pages 564–575, 2002.

[13] E. Ukkonen. On-line construction of suffix trees. *Algorithmica*, 14(3):249–260, 1995.

[14] S. V. N. Vishwanathan. *Kernel Methods: Fast Algorithms and Real Life Applications*. PhD thesis, Indian Institute of Science, Bangalore, India, November 2002.

[15] C. Watkins. Dynamic alignment kernels. In A. J. Smola, P. L. Bartlett, B. Schölkopf, and D. Schuurmans, editors, *Advances in Large Margin Classifiers*, pages 39–50, Cambridge, MA, 2000. MIT Press.